# Large-Scale Multiclass Transduction

**Thomas Gärtner**
Fraunhofer AIS.KD, 53754 Sankt Augustin, Thomas.Gaertner@ais.fraunhofer.de

**Quoc V. Le, Simon Burton, Alex J. Smola, Vishy Vishwanathan**
Statistical Machine Learning Program, NICTA and ANU, Canberra, ACT
{Quoc.Le, Simon.Burton, Alex.Smola, SVN.Vishwanathan}@nicta.com.au

## Abstract

We present a method for performing transductive inference on very large datasets. Our algorithm is based on multiclass Gaussian processes and is effective whenever the multiplication of the kernel matrix or its inverse with a vector can be computed sufficiently fast. This holds, for instance, for certain graph and string kernels. Transduction is achieved by variational inference over the unlabeled data subject to a balancing constraint.

## 1 Introduction

While obtaining labeled data remains a time and labor consuming task, acquisition and storage of unlabelled data is becoming increasingly cheap and easy. This development has driven machine learning research into exploring algorithms that make extensive use of unlabelled data at training time in order to obtain better generalization performance.

A common problem of many transductive approaches is that they scale badly with the amount of unlabeled data, which prohibits the use of massive sets of unlabeled data. Our algorithm shows improved scaling behavior, both for standard Gaussian Process classification and transduction. We perform classification on a dataset consisting of a digraph with $75,888$ vertices and $508,960$ edges. To the best of our knowledge it has so far not been possible to perform transduction on graphs of this size in reasonable time (with standard hardware). On standard data our method shows competitive or better performance.

**Existing Transductive Approaches** for SVMs use nonlinear programming [2] or EM-style iterations for binary classification [4]. Moreover, on graphs various methods for unsupervised learning have been proposed [12, 11], all of which are mainly concerned with computing the kernel matrix on training and test set jointly. Other formulations impose that the label assignment on the test set be consistent with the assumption of confident classification [8]. Yet others impose that training and test set have similar marginal distributions [4].

The present paper uses all three properties. It is particularly efficient whenever $K\alpha$ or $K^{-1}\alpha$ can be computed in linear time, where $K \in \mathbb{R}^{m \times m}$ is the kernel matrix and $\alpha \in \mathbb{R}^m$.

- We require consistency of training and test marginals. This avoids problems with overly large majority classes and small training sets.
- Kernels (or their inverses) are computed on training and test set simultaneously. On graphs this can lead to considerable computational savings.
- Self consistency of the estimates is achieved by a variational approach. This allows us to make use of Gaussian Process multiclass formulations.

## 2 Multiclass Classification

We begin with a brief overview over Gaussian Process multiclass classification [10] recast in terms of exponential families. Denote by $\mathcal{X} \times \mathcal{Y}$ with $\mathcal{Y} = \{1..n\}$ the domain of observations and labels. Moreover let $X := \{x_1, \dots, x_m\}$ and $Y := \{y_1, \dots, y_m\}$ be the set of observations. It is our goal to estimate $y|x$ via

$$p(y|x,\theta) = \exp\left(\langle \phi(x,y), \theta \rangle - g(\theta|x)\right) \text{ where } g(\theta|x) = \log \sum_{y \in \mathcal{Y}} \exp\left(\langle \phi(x,y), \theta \rangle\right). \quad (1)$$

$\phi(x,y)$ are the joint sufficient statistics of $x$ and $y$ and $g(\theta|x)$ is the log-partition function which takes care of the normalization. We impose a normal prior on $\theta$, leading to the following negative joint likelihood in $\theta$ and $Y$:

$$\mathcal{P} := -\log p(\theta, Y|X) = \sum_{i=1}^{m} \left[g(\theta|x_i) - \langle \phi(x_i, y_i), \theta \rangle\right] + \frac{1}{2\sigma^2} \|\theta\|^2 + \text{const.} \quad (2)$$

For transduction purposes $p(\theta, Y|X)$ will prove more useful than $p(\theta|Y, X)$. Note that a normal prior on $\theta$ with variance $\sigma^2 \mathbf{1}$ implies a Gaussian process on the random variable $t(x,y) := \langle \phi(x,y), \theta \rangle$ with covariance kernel

$$\text{Cov}\left[t(x,y), t(x',y')\right] = \sigma^2 \langle \phi(x,y), \phi(x',y') \rangle =: \sigma^2 k((x,y), (x',y')). \quad (3)$$

**Parametric Optimization Problem**  In the following we assume isotropy among the class labels, that is $\langle \phi(x,y), \phi(x',y') \rangle = \delta_{y,y'} \langle \phi(x), \phi(x') \rangle$ (this is not a necessary requirement for the efficiency of our algorithm, however it greatly simplifies the presentation). This allows us to decompose $\theta$ into $\theta_1, \dots, \theta_n$ such that

$$\langle \phi(x,y), \theta \rangle = \langle \phi(x), \theta_y \rangle \text{ and } \|\theta\|^2 = \sum_{y=1}^{n} \|\theta_y\|^2. \quad (4)$$

Applying the representer theorem allows us to expand $\theta$ in terms of $\phi(x_i, y_i)$ as $\theta = \sum_{i=1}^{m} \sum_{y=1}^{n} \alpha_{iy} \phi(x_i, y)$. In conjunction with (4) we have

$$\theta_y = \sum_{i=1}^{m} \alpha_{iy} \phi(x_i) \text{ where } \alpha \in \mathbb{R}^{m \times n}. \quad (5)$$

Let $\mu \in \mathbb{R}^{m \times n}$ with $\mu_{ij} = 1$ if $y_i = j$ and $\mu_{ij} = 0$ otherwise, and $K \in \mathbb{R}^{m \times m}$ with $K_{ij} = \langle \phi(x_i), \phi(x_j) \rangle$. Here joint log-likelihood (2) in terms of $\alpha$ and $K$ yields

$$\sum_{i=1}^{m} \log \sum_{y=1}^{n} \exp\left([K\alpha]_{iy}\right) - \text{tr}\,\mu^\top K\alpha + \frac{1}{2\sigma^2} \text{tr}\,\alpha^\top K\alpha + \text{const.} \quad (6)$$

Equivalently we could expand (2) in terms of $t := K\alpha$. This is commonly done in Gaussian process literature and we will use both formulations, depending on the problem we need to solve: if $K\alpha$ can be computed effectively, as is the case with string kernels [9], we use the $\alpha$-parameterization. Conversely, if $K^{-1}\alpha$ is cheap, as for example with graph kernels [7], we use the $t$-parameterization.

**Derivatives**  Second order methods such as Conjugate Gradient require the computation of derivatives of $-\log p(\theta, Y|X)$ with respect to $\theta$ in terms of $\alpha$ or $t$. Using the shorthand $\pi \in \mathbb{R}^{m \times n}$ with $\pi_{ij} := p(y = j|x_i, \theta)$ we have

$$\partial_\alpha \mathcal{P} = K(\pi - \mu + \sigma^{-2}\alpha) \text{ and } \partial_t \mathcal{P} = \pi - \mu + \sigma^{-2}K^{-1}t. \quad (7)$$

To avoid spelling out tensors of fourth order for the second derivatives (since $\alpha \in \mathbb{R}^{m \times n}$) we state the action of the latter as bilinear forms on vectors $\beta, \gamma, u, v \in \mathbb{R}^{m \times n}$. For convenience we use the "Matlab" notation of '.*' to denote element-wise multiplication of matrices:

$$\partial_\alpha^2 \mathcal{P}[\beta, \gamma] = \operatorname{tr}(K\gamma)^\top (\pi. * (K\beta)) - \operatorname{tr}(\pi. * K\gamma)^\top (\pi. * (K\beta)) + \sigma^{-2} \operatorname{tr} \gamma^\top K\beta \quad \text{(8a)}$$

$$\partial_t^2 \mathcal{P}[u, v] = \operatorname{tr} u^\top (\pi. * v) - \operatorname{tr}(\pi. * u)^\top (\pi. * v) + \sigma^{-2} \operatorname{tr} u^\top K^{-1} v. \quad \text{(8b)}$$

Let $L \cdot n$ be the computational time required to compute $K\alpha$ and $K^{-1}t$ respectively. One may check that $L = O(m)$ implies that each conjugate gradient (CG) descent step can be performed in $O(m)$ time. Combining this with rates of convergence for Newton-type or nonlinear CG solver strategies yields overall time costs in the order of $O(m \log m)$ to $O(m^2)$ worst case, a significant improvement over conventional $O(m^3)$ methods.

## 3  Transductive Inference by Variational Methods

As we are interested in transduction, the labels $Y$ (and analogously the data $X$) decompose as $Y = Y_{\text{train}} \cup Y_{\text{test}}$. To directly estimate $p(Y_{\text{test}}|X, Y_{\text{train}})$ we would need to integrating out $\theta$, which is usually intractable. Instead, we now aim at estimating the mode of $p(\theta|X, Y_{\text{train}})$ by variational means. With the KL-divergence $D$ and an arbitrary distribution $q$ the well-known bound (see e.g. [5])

$$-\log p(\theta|X, Y_{\text{train}}) \leq -\log p(\theta|X, Y_{\text{train}}) + D(q(Y_{\text{test}}) \| p(Y_{\text{test}}|X, Y_{\text{train}}, \theta)) \quad \text{(9)}$$

$$= -\sum_{Y_{\text{test}}} \left( \log p(Y_{\text{test}}, \theta|X, Y_{\text{train}}) - \log q(Y_{\text{test}}) \right) q(Y_{\text{test}}) \quad \text{(10)}$$

holds. This bound (10) can be minimized with respect to $\theta$ and $q$ in an iterative fashion. The key trick is that while using a factorizing approximation for $q$ we restrict the latter to distributions which satisfy balancing constraints. That is, we require them to yield marginals on the unlabeled data which are comparable with the labeled observations.

**Decomposing the Variational Bound**  To simplify (10) observe that

$$p(Y_{\text{test}}, \theta|X, Y_{\text{train}}) = p(Y_{\text{train}}, Y_{\text{test}}, \theta|X)/p(Y_{\text{train}}|X). \quad \text{(11)}$$

In other words, the first term in (10) equals (6) up to a constant independent of $\theta$ or $Y_{\text{test}}$. With $q_{ij} := q(y_i = j)$ we define $\mu_{ij}(q) = q_{ij}$ for all $i > m_{\text{train}}$ and $\mu_{ij}(q) = 1$ if $y_i = 1$ and 0 otherwise for all $i \leq m_{\text{train}}$. In other words, we are taking the expectation in $\mu$ over all unobserved labels $Y_{\text{test}}$ with respect to the distribution $q(Y_{\text{test}})$. We have

$$\sum_{Y_{\text{test}}} q(Y_{\text{test}}) \log p(Y_{\text{test}}, \theta|X, Y_{\text{train}})$$

$$= \sum_{i=1}^m \log \sum_{j=1}^n \exp\left([K\alpha]_{ij}\right) - \operatorname{tr} \mu(q)^\top K\alpha + \frac{1}{2\sigma^2} \operatorname{tr} \alpha^\top K\alpha + \text{const.} \quad \text{(12)}$$

For fixed $q$ the optimization over $\theta$ proceeds as in Section 2. Next we discuss $q$.

**Optimization over $q$**  The second term in (10) is the negative entropy of $q$. Since $q$ factorizes we have

$$\sum_{Y_{\text{test}}} q(Y_{\text{test}}) \log q(Y_{\text{test}}) = \sum_{i=m_{\text{train}}+1}^m q_{ij} \log q_{ij}. \quad \text{(13)}$$

It is unreasonable to assume that $q$ may be chosen freely from all factorizing distributions (the latter would lead to a straightforward EM algorithm for transductive inference): if we observe a certain distribution of labels on the training set, e.g., for binary classification we see 45% positive and 55% negative labels, then it is very unlikely that the label distribution on the test set deviates significantly. Hence we should make use of this information.

If $m \gg m_{\text{train}}$, however, a naive application of the variational bound can lead to cases where $q$ is concentrated on one class — the increase in likelihood for a resulting very simple classifier completely outweighs any balancing constraints implicit in the data. This is confirmed by experimental results. It is, incidentally, also the reason why SVM transduction optimization codes [4] impose a balancing constraint on the assignment of test labels. We impose the following conditions:

$$r_j^- \leq \sum_{i=m_{\text{train}}+1}^{m} q_{ij} \leq r_j^+ \text{ for all } j \in \mathcal{Y} \text{ and } \sum_{j=1}^{n} q_{ij} = 1 \text{ for all } i \in \{m_{\text{train}}..m\}.$$

Here the constraints $r_j^- = p_{\text{emp}}(y = j) - \epsilon$ and $r_j^+ = p_{\text{emp}}(y = j) + \epsilon$ are chosen such as to correspond to confidence intervals given by finite sample size tail bounds. In other words we set $p_{\text{emp}}(y = j) = m_{\text{train}}^{-1} \sum_{i=1}^{m_{\text{train}}} \{y_i = j\}$ and $\epsilon$ such as to satisfy

$$\Pr \left\{ \left| m_{\text{train}}^{-1} \sum_{i=1}^{m_{\text{train}}} \xi_i - m_{\text{test}}^{-1} \sum_{i=1}^{m_{\text{test}}} \xi_i' \right| > \epsilon \right\} \leq \delta \tag{14}$$

for iid $\{0, 1\}$ random variables $\xi_i$ and $\xi_i'$ with mean $p$. This is a standard ghost-sample inequality. It follows directly from [3, Eq. (2.7)] after application of a union bound over the class labels that $\epsilon \leq \sqrt{\log(2n/\delta)m/(2m_{\text{train}}m_{\text{test}})}$.

## 4  Graphs, Strings and Vectors

We now discuss the two main applications where computational savings can be achieved: graphs and strings. In the case of graphs, the advantage arises from the fact that $K^{-1}$ is sparse, whereas for texts we can use fast string kernels [9] to compute $K\alpha$ in linear time.

**Graphs** Denote by $G(V, E)$ the graph given by vertices $V$ and edges $E$ where each edge is a set of two vertices. Then $W \in \mathbb{R}^{|V| \times |V|}$ denotes the adjacency matrix of the graph, where $W_{ij} > 0$ only if edge $\{i, j\} \in E$. We assume that the graph $G$, and thus also the adjacency matrix $W$, is sparse. Now denote by $\mathbf{1}$ the identity matrix and by $D$ the diagonal matrix of vertex degrees, i.e., $D_{ii} = \sum_j W_{ij}$. Then the graph Laplacian and the normalized graph Laplacian of $G$ are given by

$$L := D - W \quad \text{and} \quad \tilde{L} := \mathbf{1} - D^{-\frac{1}{2}}WD^{-\frac{1}{2}}, \tag{15}$$

respectively. Many kernels $K$ (or their inverse) on $G$ are given by low-degree polynomials of the Laplacian or the adjacency matrix of $G$, such as the following:

$$K = \sum_{i=1}^{l} c_i W^{2i}, K = \prod_{i=1}^{l} (\mathbf{1} - c_i \tilde{L}), \text{ or } K^{-1} = \tilde{L} + \epsilon \mathbf{1}. \tag{16}$$

In all three cases we assumed $c_i, \epsilon \geq 0$ and $l \in \mathbb{N}$. The first kernel arises from an $l$-step random walk, the third case is typically referred to as regularized graph Laplacian. In these cases $K\alpha$ or $K^{-1}t$ can be computed using $L = l(|V| + |E|)$ operations. This means that if the average degree of the graph does not increase with the number of observations, $L = O(m)$ as $m = |V|$ for inference on graphs.

**From Graphs to Graphical Models** Graphs are one of the examples where transduction actually improves computational cost: Assume that we are given the inverse kernel matrix $K^{-1}$ on training and test set and we wish to perform induction only. In this case we need to compute the kernel matrix (or its inverse) restricted to the training set. Let $K^{-1} = \begin{bmatrix} A & B \\ B^\top & C \end{bmatrix}$, then the upper left hand corner (representing the training set part only) of

$K$ is given by the Schur complement $\left(A - B^\top C^{-1} B\right)^{-1}$. Computing the latter is costly. Moreover, neither the Schur complement nor its inverse are typically sparse.

Here we have a nice connection between graphical models and graph kernels. Assume that $t$ is a normal random variable with conditional independence properties. In this case the inverse covariance matrix has nonzero entries only for variables with a direct dependency structure. This follows directly from an application of the Clifford-Hammersley theorem to Gaussian random variables [6]. In other words, if we are given a graphical model of normal random variables, their conditional independence structure is reflected by $K^{-1}$.

In the same way as in graphical models marginalization may induce dependencies, computing the kernel matrix on the training set only, may lead to dense matrices, even when the inverse kernel on training and test data combined is sparse. The bottom line is there are cases where it is computationally cheaper to take both training and test set into account and optimize over a larger set of variables rather than dealing with a smaller dense matrix.

**Strings:** Efficient computation of string kernels using suffix trees was described in [9]. In particular, it was observed that expansions of the form $\sum_{i=1}^{m} \alpha_i k(x_i, x)$ can be evaluated in linear time in the length of $x$, provided some preprocessing for the coefficients $\alpha$ and observations $x_i$ is performed. This preprocessing is independent of $x$ and can be computed in $O(\sum_i |x_i|)$ time. The efficient computation scheme covers all kernels of type

$$k(x, x') = \sum_s w_s \#_s(x) \#_s(x') \tag{17}$$

for arbitrary $w_s \geq 0$. Here, $\#_s(x)$ denotes the number of occurrences of $s$ in $x$ and the sum is carried out over all substrings of $x$. This means that computation time for evaluating $K\alpha$ is again $O(\sum_i |x_i|)$ as we need to evaluate the kernel expansion for all $x \in X$. Since the average string length is independent of $m$ this yields an $O(m)$ algorithm for $K\alpha$.

**Vectors:** If $k(x, x') = \phi(x)^\top \phi(x')$ and $\phi(x) \in \mathbb{R}^d$ for $d \ll m$, it is possible to carry out matrix vector multiplications in $O(md)$ time. This is useful for cases where we have a sparse matrix with a small number of low-rank updates (e.g. from low rank dense fill-ins).

## 5 Optimization

**Optimization in $\alpha$ and $t$:** $\mathcal{P}$ is convex in $\alpha$ (and in $t$ since $t = K\alpha$). This means that a combination of Conjugate-Gradient and Newton-Raphson (NR) can be used for optimization.

- Compute updates $\alpha \longleftarrow \alpha - \eta \partial_\alpha^2 \mathcal{P}^{-1} \partial_\alpha \mathcal{P}$ via
  - Solve the linear system approximately by Conjugate Gradient iterations.
  - Find optimal $\eta$ by line search.
- Repeat until the norm of the gradient is sufficiently small.

Key is the fact that the arising linear system is only solved approximately, which can be done using very few CG iterations. Since each of them is $O(m)$ for fast kernel-vector computations the overall cost is a sub-quadratic function of $m$.

**Optimization in $q$** is somewhat less straightforward: we need to find the optimal $q$ in terms of KL-divergence subject to the marginal constraint. Denote by $\tau$ the part of $K\alpha$ pertaining to test data, or more formally $\tau \in \mathbb{R}^{m_\text{test} \times n}$ with $\tau_{ij} = [K\alpha]_{i+m_\text{train}, j}$. We have:

$$\underset{q}{\text{minimize}} \ \ \text{tr} \, q^\top \tau + \sum_{i,j} q_{ij} \log q_{ij} \tag{18}$$

$$\text{subject to } q_j^- \leq \sum_i q_{ij} \leq q_j^+, q_{ij} \geq 0 \text{ and } \sum_i q_{li} = 1 \text{ for all } j \in \mathcal{Y}, l \in \{1..m_\text{test}\}$$

Table 1: Error rates on some benchmark datasets (mostly from UCI). The last column is the error rates reported in [1]

| DATASET | #INST | #ATTR | IND. GP | TRANSD. GP | S$^3$VMMIP |
|---|---|---|---|---|---|
| cancer | 699 | 9 | 3.4%±4.1% | 2.1%±4.7% | 3.4% |
| cancer (progn.) | 569 | 30 | 6.1%±3.7% | 6.0%±3.7% | 3.3% |
| heart (cleave.) | 297 | 13 | 15.0%±5.6% | 13.0%±6.3% | 16.0% |
| housing | 506 | 13 | 7.0%±1.0% | 6.8%±0.9% | 15.1% |
| ionosphere | 351 | 34 | 8.6%±6.3% | 6.1%±3.4% | 10.6% |
| pima | 769 | 8 | 19.6%±8.1% | 17.6%±8.0% | 22.2% |
| sonar | 208 | 60 | 10.5%±5.1% | 8.6%±3.4% | 21.9% |
| glass | 214 | 10 | 20.5%±1.6% | 17.3%±4.5% | — |
| wine | 178 | 13 | 19.4%±5.7% | 15.6%±4.2% | — |
| tictactoe | 958 | 9 | 3.9%±0.7% | 3.3%±0.6% | — |
| cmc | 1473 | 10 | 32.5%±7.1% | 28.9%±7.5% | — |
| USPS | 9298 | 256 | 5.9% | 4.8% | —[1] |

This is a convex optimization problem. Using Lagrange multipliers one can show that $q$ needs to satisfy $q_{ij} = \exp(-\tau_{ij})b_i c_j$ where $b_i, c_j \geq 0$. Solving for $\sum_j^n q_{ij} = 1$ yields $q_{ij} = \frac{\exp(-\tau_{ij})c_j}{\sum_{l=1}^n \exp(-\tau_{il})c_l}$. This means that instead of an optimization problem in $m_{\text{test}} \times n$ variables we now only need to optimize over $n$ variables subject to $2n$ constraints.

Note that the exact matching constraint where $q_i^+ = q_i^-$ amounts to a maximum likelihood problem for a shifted exponential family model where $q_{ij} = \exp(\tau_{ij})\exp(\gamma_i - g_j(\gamma_i))$. It can be shown that the approximate matching problem is equivalent to a maximum a posteriori optimization problem using the norm dual to expectation constraints on $q_{ij}$. We are currently working on extending this setting

In summary, the optimization now only depends on $n$ variables. It can be solved by standard second order methods. As initialization we choose $\gamma_i$ such that the per class averages match the marginal constraint while ignoring the per sample balance. After that a small number Newton steps suffices for optimization.

## 6   Experiments

Unfortunately, we are not aware of other multiclass transductive learning algorithms. To still be able to compare our approach to other transductive learning algorithms we performed experiments on some benchmark datasets. To investigate the performance of our algorithm in classifying vertices of a graph, we choose the WebKB dataset.

**Benchmark datasets** Table 1 reports results on some benchmark datasets. To be able to compare the error rates of the transductive multiclass Gaussian Process classifier proposed in this paper, we also report error rates from [2] and an inductive multiclass Gaussian Process classifier. The reported error rates are for 10-fold crossvalidations. Parameters were chosen by crossvalidation inside the training folds.

**Graph Mining** To illustrate the effectiveness of our approach on graphs we performed experiments on the well known WebKB dataset. This dataset consists of 8275 webpages classified into 7 classes. Each webpage contains textual content and/or links to other webpages. As we are using this dataset to evaluate our graph mining algorithm, we ignore the text on each webpage and consider the dataset as a labelled directed graph. To have the data

Table 2: Results on WebKB for 'inverse' 10-fold crossvalidation

| DATASET | $|V|$ | $|E|$ | ERROR | DATASET | $|V|$ | $|E|$ | ERROR |
|---|---|---|---|---|---|---|---|
| Cornell | 867 | 1793 | 10% | Misc | 4113 | 4462 | 66% |
| Texas | 827 | 1683 | 8% | all | 8275 | 14370 | 53% |
| Washington | 1205 | 2368 | 10% | Universities | 4162 | 9591 | 12% |
| Wisconsin | 1263 | 3678 | 15% | | | | |

set as large as possible, we did not remove any webpages, opposed to most other work.

Table 2 reports the results of our algorithm on different subsets of the WebKB data as well as on the full data. We use the co-linkage graph and report results for 'inverse' 10-fold stratified crossvalidations, i.e., we use 1 fold as training data and 9 folds as test data. Parameters are the same for all reported experiments and were found by experimenting with a few parametersets on the 'Cornell' subset only. It turned out that the class membership probabilities are not well-calibrated on this dataset. To overcome this, we predict on the test set as follows: For each class the instances that are most likely to be in this class are picked (if they haven't been picked for a class with lower index) such that the fraction of instances assigned to this class is the same on the training and test set. We will investigate the reason for this in future work.

The setting most similar to ours is probably the one described in [11]. Although a directed graph approach outperforms there an undirected approach, we resorted to kernels for undirected graphs, as those are computationally more attractive. We will investigate computationally attractive digraph kernels in future work and expect similar benefits as reported by [11]. Though we are using more training data than [11] we are also considering a more difficult learning problem (multiclass without removing various instances). To investigate the behaviour of our algorithm with less training data, we performed a 20-fold inverse crossvalidation on the 'wisconsin' subset and observed an error rate of 17% there.

To further strengthen our results and show that the runtime performance of our algorithm is sufficient for classifying the vertices of massive graphs, we also performed initial experiments on the Epinions dataset collected by Mathew Richardson and Pedro Domingos. The dataset is a social network consisting of $75,888$ people connected by $508,960$ 'trust' edges. Additionally the dataset comes with a list of 185 'topreviewers' for 25 topic areas. We tried to predict these but only got 12% of the topreviewers correct. As we are not aware of any predictive results on this task, we suppose this low accuracy is inherent to this task. However, the experiments show that the algorithm can be run on very large graph datasets.

## 7  Discussion and Extensions

We presented an efficient method for performing transduction on multiclass estimation problems with Gaussian Processes. It performs particularly well whenever the kernel matrix has special numerical properties which allow fast matrix vector multiplication. That said, also on standard dense problems we observed very good improvements (typically a 10% reduction of the training error) over standard induction.

**Structured Labels and Conditional Random Fields** are a clear area where to extend the transductive setting. The key obstacle to overcome in this context is to find a suitable marginal distribution: with increasing structure of the labels the confidence bounds per subclass decrease dramatically. A promising strategy is to use only partial marginals on maximal cliques and enforce them directly similarly to an unconditional Markov network.

**Applications to Document Analysis** require efficient small-memory-footprint suffix tree implementations. We are currently working on this, which will allow GP classification to perform estimation on large document collections. We believe it will be possible to use out-of-core storage in conjunction with annotation to work on sequences of $10^8$ characters.

**Other Marginal Constraints** than matching marginals are worth exploring. In particular, constraints derived from exchangeable distributions such as those used by Latent Dirichlet Allocation are a promising area to consider. This may also lead to connections between GP classification and clustering.

**Sparse** $O(m^{1.3})$ **Solvers for Graphs** have recently been proposed by the theoretical computer science community. It is worthwhile exploring their use for inference on graphs.

**Acknowledgements** The authors thank Mathew Richardson and Pedro Domingos for collecting the Epinions data and Deepayan Chakrabarti and Christos Faloutsos for providing a preprocessed version. Parts of this work were carried out when TG was visiting NICTA. National ICT Australia is funded through the Australian Government's *Backing Australia's Ability* initiative, in part through the Australian Research Council. This work was supported by grants of the ARC and by the Pascal Network of Excellence.

## Footnotes

[1]In [2] only subsets of USPS were considered due to the size of this problem.

# References

[1] K. Bennett. Combining support vector and mathematical programming methods for classification. In *Advances in Kernel Methods - -Support Vector Learning*, pages 307 – 326. MIT Press, 1998.

[2] K. Bennett. Combining support vector and mathematical programming methods for induction. In B. Schölkopf, C. J. C. Burges, and A. J. Smola, editors, *Advances in Kernel Methods - -SV Learning*, pages 307 – 326, Cambridge, MA, 1999. MIT Press.

[3] W. Hoeffding. Probability inequalities for sums of bounded random variables. *Journal of the American Statistical Association*, 58:13 – 30, 1963.

[4] T. Joachims. *Learning to Classify Text Using Support Vector Machines: Methods, Theory, and Algorithms*. The Kluwer International Series In Engineering And Computer Science. Kluwer Academic Publishers, Boston, May 2002. ISBN 0 - 7923 - 7679-X.

[5] M. I. Jordan, Z. Ghahramani, Tommi S. Jaakkola, and L. K. Saul. An introduction to variational methods for graphical models. *Machine Learning*, 37(2):183 – 233, 1999.

[6] S. L. Lauritzen. *Graphical Models*. Oxford University Press, 1996.

[7] A. J. Smola and I. R. Kondor. Kernels and regularization on graphs. In B. Schölkopf and M. K. Warmuth, editors, *Proceedings of the Annual Conference on Computational Learning Theory*, Lecture Notes in Computer Science. Springer, 2003.

[8] V. Vapnik. *Statistical Learning Theory*. John Wiley and Sons, New York, 1998.

[9] S. V. N. Vishwanathan and A. J. Smola. Fast kernels for string and tree matching. In K. Tsuda, B. Schölkopf, and J.P. Vert, editors, *Kernels and Bioinformatics*, Cambridge, MA, 2004. MIT Press.

[10] C. K. I. Williams and D. Barber. Bayesian classification with Gaussian processes. *IEEE Transactions on Pattern Analysis and Machine Intelligence PAMI*, 20(12):1342 – 1351, 1998.

[11] D. Zhou, J. Huang, and B. Schölkopf. Learning from labeled and unlabeled data on a directed graph. In *International Conference on Machine Learning*, 2005.

[12] X. Zhu, J. Lafferty, and Z. Ghahramani. Semi-supervised learning using gaussian fields and harmonic functions. In *International Conference on Machine Learning ICML'03*, 2003.
